# Deflation Methods for Sparse PCA

**Lester Mackey**
Computer Science Division
University of California, Berkeley
Berkeley, CA 94703

## Abstract

In analogy to the PCA setting, the sparse PCA problem is often solved by iteratively alternating between two subtasks: cardinality-constrained rank-one variance maximization and matrix deflation. While the former has received a great deal of attention in the literature, the latter is seldom analyzed and is typically borrowed without justification from the PCA context. In this work, we demonstrate that the standard PCA deflation procedure is seldom appropriate for the sparse PCA setting. To rectify the situation, we first develop several deflation alternatives better suited to the cardinality-constrained context. We then reformulate the sparse PCA optimization problem to explicitly reflect the maximum *additional* variance objective on each round. The result is a generalized deflation procedure that typically outperforms more standard techniques on real-world datasets.

## 1 Introduction

Principal component analysis (PCA) is a popular change of variables technique used in data compression, predictive modeling, and visualization. The goal of PCA is to extract several principal components, linear combinations of input variables that together best account for the variance in a data set. Often, PCA is formulated as an eigenvalue decomposition problem: each eigenvector of the sample covariance matrix of a data set corresponds to the *loadings* or coefficients of a principal component. A common approach to solving this partial eigenvalue decomposition is to iteratively alternate between two subproblems: rank-one variance maximization and matrix deflation. The first subproblem involves finding the maximum-variance loadings vector for a given sample covariance matrix or, equivalently, finding the leading eigenvector of the matrix. The second involves modifying the covariance matrix to eliminate the influence of that eigenvector.

A primary drawback of PCA is its lack of sparsity. Each principal component is a linear combination of all variables, and the loadings are typically non-zero. Sparsity is desirable as it often leads to more interpretable results, reduced computation time, and improved generalization. Sparse PCA [8, 3, 16, 17, 6, 18, 1, 2, 9, 10, 12] injects sparsity into the PCA process by searching for "pseudo-eigenvectors", sparse loadings that explain a maximal amount variance in the data.

In analogy to the PCA setting, many authors attempt to solve the sparse PCA problem by iteratively alternating between two subtasks: cardinality-constrained rank-one variance maximization and matrix deflation. The former is an NP-hard problem, and a variety of relaxations and approximate solutions have been developed in the literature [1, 2, 9, 10, 12, 16, 17]. The latter subtask has received relatively little attention and is typically borrowed without justification from the PCA context. In this work, we demonstrate that the standard PCA deflation procedure is seldom appropriate for the sparse PCA setting. To rectify the situation, we first develop several heuristic deflation alternatives with more desirable properties. We then reformulate the sparse PCA optimization problem to explicitly reflect the maximum *additional* variance objective on each round. The result is a generalized deflation procedure that typically outperforms more standard techniques on real-world datasets.

The remainder of the paper is organized as follows. In Section 2 we discuss matrix deflation as it relates to PCA and sparse PCA. We examine the failings of typical PCA deflation in the sparse setting and develop several alternative deflation procedures. In Section 3, we present a reformulation of the standard iterative sparse PCA optimization problem and derive a generalized deflation procedure to solve the reformulation. Finally, in Section 4, we demonstrate the utility of our newly derived deflation techniques on real-world datasets.

**Notation**

$I$ is the identity matrix. $\mathbb{S}_+^p$ is the set of all symmetric, positive semidefinite matrices in $\mathbb{R}^{p \times p}$. $\mathbf{Card}(x)$ represents the cardinality of or number of non-zero entries in the vector $x$.

## 2 Deflation methods

A *matrix deflation* modifies a matrix to eliminate the influence of a given eigenvector, typically by setting the associated eigenvalue to zero (see [14] for a more detailed discussion). We will first discuss deflation in the context of PCA and then consider its extension to sparse PCA.

### 2.1 Hotelling's deflation and PCA

In the PCA setting, the goal is to extract the $r$ leading eigenvectors of the sample covariance matrix, $A_0 \in \mathbb{S}_+^p$, as its eigenvectors are equivalent to the loadings of the first $r$ principal components. Hotelling's deflation method [11] is a simple and popular technique for sequentially extracting these eigenvectors. On the $t$-th iteration of the deflation method, we first extract the leading eigenvector of $A_{t-1}$,

$$x_t = \underset{x:x^T x=1}{\operatorname{argmax}} \, x^T A_{t-1} x \tag{1}$$

and we then use Hotelling's deflation to annihilate $x_t$:

$$A_t = A_{t-1} - x_t x_t^T A_{t-1} x_t x_t^T. \tag{2}$$

The deflation step ensures that the $t+1$-st leading eigenvector of $A_0$ is the leading eigenvector of $A_t$. The following proposition explains why.

**Proposition 2.1.** *If $\lambda_1 \geq \ldots \geq \lambda_p$ are the eigenvalues of $A \in \mathbb{S}_+^p$, $x_1, \ldots, x_p$ are the corresponding eigenvectors, and $\hat{A} = A - x_j x_j^T A x_j x_j^T$ for some $j \in 1, \ldots, p$, then $\hat{A}$ has eigenvectors $x_1, \ldots, x_p$ with corresponding eigenvalues $\lambda_1, \ldots, \lambda_{j-1}, 0, \lambda_{j+1}, \ldots, \lambda_p$.*

PROOF.

$$\hat{A}x_j = Ax_j - x_j x_j^T A x_j x_j^T x_j = Ax_j - x_j x_j^T A x_j = \lambda_j x_j - \lambda_j x_j = 0x_j.$$
$$\hat{A}x_i = Ax_i - x_j x_j^T A x_j x_j^T x_i = Ax_i - 0 = \lambda_i x_i, \forall i \neq j.$$

$\square$

Thus, Hotelling's deflation preserves all eigenvectors of a matrix and annihilates a selected eigenvalue while maintaining all others. Notably, this implies that Hotelling's deflation preserves positive-semidefiniteness. In the case of our iterative deflation method, annihilating the $t$-th leading eigenvector of $A_0$ renders the $t+1$-st leading eigenvector dominant in the next round.

### 2.2 Hotelling's deflation and sparse PCA

In the sparse PCA setting, we seek $r$ sparse loadings which together capture the maximum amount of variance in the data. Most authors [1, 9, 16, 12] adopt the additional constraint that the loadings be produced in a sequential fashion. To find the first such "pseudo-eigenvector", we can consider a cardinality-constrained version of Eq. (1):

$$x_1 = \underset{x:x^T x=1, \mathbf{Card}(x) \leq k_1}{\operatorname{argmax}} \, x^T A_0 x. \tag{3}$$

That leaves us with the question of how to best extract subsequent pseudo-eigenvectors. A common approach in the literature [1, 9, 16, 12] is to borrow the iterative deflation method of the PCA setting. Typically, Hotelling's deflation is utilized by substituting an extracted pseudo-eigenvector for a true eigenvector in the deflation step of Eq. (2). This substitution, however, is seldom justified, for the properties of Hotelling's deflation, discussed in Section 2.1, depend crucially on the use of a true eigenvector.

To see what can go wrong when Hotelling's deflation is applied to a non-eigenvector, consider the following example.

**Example.** Let $C = \begin{pmatrix} 2 & 1 \\ 1 & 1 \end{pmatrix}$, a $2 \times 2$ matrix. The eigenvalues of $C$ are $\lambda_1 = 2.6180$ and $\lambda_2 = .3820$. Let $x = (1, 0)^T$, a sparse pseudo-eigenvector, and $\hat{C} = C - xx^T Cxx^T$, the corresponding deflated matrix. Then $\hat{C} = \begin{pmatrix} 0 & 1 \\ 1 & 1 \end{pmatrix}$ with eigenvalues $\hat{\lambda_1} = 1.6180$ and $\hat{\lambda_2} = -.6180$. Thus, Hotelling's deflation does not in general preserve positive-semidefiniteness when applied to a non-eigenvector.

That $\mathbb{S}_+^p$ is not closed under pseudo-eigenvector Hotelling's deflation is a serious failing, for most iterative sparse PCA methods assume a positive-semidefinite matrix on each iteration. A second, related shortcoming of pseudo-eigenvector Hotelling's deflation is its failure to render a pseudo-eigenvector orthogonal to a deflated matrix. If $A$ is our matrix of interest, $x$ is our pseudo-eigenvector with variance $\lambda = x^T Ax$, and $\hat{A} = A - xx^T Axx^T$ is our deflated matrix, then $\hat{A}x = Ax - xx^T Axx^T x = Ax - \lambda x$ is zero iff $x$ is a true eigenvector. Thus, even though the "variance" of $x$ w.r.t. $\hat{A}$ is zero ($x^T \hat{A}x = x^T Ax - x^T xx^T Axx^T x = \lambda - \lambda = 0$), "covariances" of the form $y^T \hat{A}x$ for $y \neq x$ may still be non-zero. This violation of the Cauchy-Schwarz inequality betrays a lack of positive-semidefiniteness and may encourage the reappearance of $x$ as a component of future pseudo-eigenvectors.

## 2.3 Alternative deflation techniques

In this section, we will attempt to rectify the failings of pseudo-eigenvector Hotelling's deflation by considering several alternative deflation techniques better suited to the sparse PCA setting. Note that any deflation-based sparse PCA method (e.g. [1, 9, 16, 12]) can utilize any of the deflation techniques discussed below.

### 2.3.1 Projection deflation

Given a data matrix $Y \in \mathbb{R}^{n \times p}$ and an arbitrary unit vector in $x \in \mathbb{R}^p$, an intuitive way to remove the contribution of $x$ from $Y$ is to project $Y$ onto the orthocomplement of the space spanned by $x$: $\hat{Y} = Y(I - xx^T)$. If $A$ is the sample covariance matrix of $Y$, then the sample covariance of $\hat{Y}$ is given by $\hat{A} = (I - xx^T)A(I - xx^T)$, which leads to our formulation for projection deflation:

**Projection deflation**
$$A_t = A_{t-1} - x_t x_t^T A_{t-1} - A_{t-1} x_t x_t^T + x_t x_t^T A_{t-1} x_t x_t^T = (I - x_t x_t^T)A_{t-1}(I - x_t x_t^T) \quad (4)$$

Note that when $x_t$ is a true eigenvector of $A_{t-1}$ with eigenvalue $\lambda_t$, projection deflation reduces to Hotelling's deflation:
$$A_t = A_{t-1} - x_t x_t^T A_{t-1} - A_{t-1} x_t x_t^T + x_t x_t^T A_{t-1} x_t x_t^T$$
$$= A_{t-1} - \lambda_t x_t x_t^T - \lambda_t x_t x_t^T + \lambda_t x_t x_t^T$$
$$= A_{t-1} - x_t x_t^T A_{t-1} x_t x_t^T.$$

However, in the general case, when $x_t$ is not a true eigenvector, projection deflation maintains the desirable properties that were lost to Hotelling's deflation. For example, positive-semidefiniteness is preserved:
$$\forall y, y^T A_t y = y^T (I - x_t x_t^T)A_{t-1}(I - x_t x_t^T)y = z^T A_{t-1} z$$
where $z = (I - x_t x_t^T)y$. Thus, if $A_{t-1} \in \mathbb{S}_+^p$, so is $A_t$. Moreover, $A_t$ is rendered left and right orthogonal to $x_t$, as $(I - x_t x_t^T)x_t = x_t - x_t = 0$ and $A_t$ is symmetric. Projection deflation therefore annihilates all covariances with $x_t$: $\forall v, v^T A_t x_t = x_t^T A_t v = 0$.

### 2.3.2 Schur complement deflation

Since our goal in matrix deflation is to eliminate the influence, as measured through variance and covariances, of a newly discovered pseudo-eigenvector, it is reasonable to consider the conditional variance of our data variables given a pseudo-principal component. While this conditional variance is non-trivial to compute in general, it takes on a simple closed form when the variables are normally distributed. Let $x \in \mathbb{R}^p$ be a unit vector and $W \in \mathbb{R}^p$ be a Gaussian random vector, representing the joint distribution of the data variables. If $W$ has covariance matrix $\Sigma$, then $(W, Wx)$ has covariance matrix $V = \begin{pmatrix} \Sigma & \Sigma x \\ x^T \Sigma & x^T \Sigma x \end{pmatrix}$, and $Var(W|Wx) = \Sigma - \frac{\Sigma x x^T \Sigma}{x^T \Sigma x}$ whenever $x^T \Sigma x \neq 0$ [15].

That is, the conditional variance is the Schur complement of the vector variance $x^T \Sigma x$ in the full covariance matrix $V$. By substituting sample covariance matrices for their population counterparts, we arrive at a new deflation technique:

**Schur complement deflation**

$$A_t = A_{t-1} - \frac{A_{t-1} x_t x_t^T A_{t-1}}{x_t^T A_{t-1} x_t} \tag{5}$$

Schur complement deflation, like projection deflation, preserves positive-semidefiniteness. To see this, suppose $A_{t-1} \in \mathbb{S}_+^p$. Then, $\forall v, v^T A_t v = v^T A_{t-1} v - \frac{v^T A_{t-1} x_t x_t^T A_{t-1} v}{x_t^T A_{t-1} x_t} \geq 0$ as $v^T A_{t-1} v x_t^T A_{t-1} x_t - (v^T A_{t-1} x_t)^2 \geq 0$ by the Cauchy-Schwarz inequality and $x_t^T A_{t-1} x_t \geq 0$ as $A_{t-1} \in \mathbb{S}_+^p$.

Furthermore, Schur complement deflation renders $x_t$ left and right orthogonal to $A_t$, since $A_t$ is symmetric and $A_t x_t = A_{t-1} x_t - \frac{A_{t-1} x_t x_t^T A_{t-1} x_t}{x_t^T A_{t-1} x_t} = A_{t-1} x_t - A_{t-1} x_t = 0$.

Additionally, Schur complement deflation reduces to Hotelling's deflation when $x_t$ is an eigenvector of $A_{t-1}$ with eigenvalue $\lambda_t \neq 0$:

$$\begin{aligned}
A_t &= A_{t-1} - \frac{A_{t-1} x_t x_t^T A_{t-1}}{x_t^T A_{t-1} x_t} \\
&= A_{t-1} - \frac{\lambda_t x_t x_t^T \lambda_t}{\lambda_t} \\
&= A_{t-1} - x_t x_t^T A_{t-1} x_t x_t^T.
\end{aligned}$$

While we motivated Schur complement deflation with a Gaussianity assumption, the technique admits a more general interpretation as a column projection of a data matrix. Suppose $Y \in \mathbb{R}^{n \times p}$ is a mean-centered data matrix, $x \in \mathbb{R}^p$ has unit norm, and $\hat{Y} = (I - \frac{Y x x^T Y^T}{||Yx||^2})Y$, the projection of the columns of $Y$ onto the orthocomplement of the space spanned by the pseudo-principal component, $Yx$. If $Y$ has sample covariance matrix $A$, then the sample covariance of $\hat{Y}$ is given by $\hat{A} = \frac{1}{n} Y^T (I - \frac{Y x x^T Y^T}{||Yx||^2})^T (I - \frac{Y x x^T Y^T}{||Yx||^2})Y = \frac{1}{n} Y^T (I - \frac{Y x x^T Y^T}{||Yx||^2})Y = A - \frac{A x x^T A}{x^T A x}$.

### 2.3.3 Orthogonalized deflation

While projection deflation and Schur complement deflation address the concerns raised by performing a single deflation in the non-eigenvector setting, new difficulties arise when we attempt to sequentially deflate a matrix with respect to a *series* of non-orthogonal pseudo-eigenvectors.

Whenever we deal with a sequence of non-orthogonal vectors, we must take care to distinguish between the variance explained by a vector and the *additional* variance explained, given all previous vectors. These concepts are equivalent in the PCA setting, as true eigenvectors of a matrix are orthogonal, but, in general, the vectors extracted by sparse PCA will not be orthogonal. The additional variance explained by the $t$-th pseudo-eigenvector, $x_t$, is equivalent to the variance explained by the component of $x_t$ orthogonal to the space spanned by all previous pseudo-eigenvectors, $q_t = x_t - \mathcal{P}_{t-1} x_t$, where $\mathcal{P}_{t-1}$ is the orthogonal projection onto the space spanned by $x_1, \ldots, x_{t-1}$. On each deflation step, therefore, we only want to eliminate the variance associated with $q_t$. Annihilating the full vector $x_t$ will often lead to "double counting" and could re-introduce components parallel to previously annihilated vectors. Consider the following example:

**Example.** Let $C_0 = I$. If we apply projection deflation w.r.t. $x_1 = (\frac{\sqrt{2}}{2}, \frac{\sqrt{2}}{2})^T$, the result is $C_1 = \begin{pmatrix} \frac{1}{2} & -\frac{1}{2} \\ -\frac{1}{2} & \frac{1}{2} \end{pmatrix}$, and $x_1$ is orthogonal to $C_1$. If we next apply projection deflation to $C_1$ w.r.t. $x_2 = (1,0)^T$, the result, $C_2 = \begin{pmatrix} 0 & 0 \\ 0 & \frac{1}{2} \end{pmatrix}$, is no longer orthogonal to $x_1$.

The authors of [12] consider this issue of non-orthogonality in the context of Hotelling's deflation. Their modified deflation procedure is equivalent to Hotelling's deflation (Eq. (2)) for $t = 1$ and can be easily expressed in terms of a running Gram-Schmidt decomposition for $t > 1$:

**Orthogonalized Hotelling's deflation (OHD)**

$$q_t = \frac{(I - Q_{t-1}Q_{t-1}^T)x_t}{||(I - Q_{t-1}Q_{t-1}^T)x_t||} \tag{6}$$
$$A_t = A_{t-1} - q_t q_t^T A_{t-1} q_t q_t^T$$

where $q_1 = x_1$, and $q_1, \ldots, q_{t-1}$ form the columns of $Q_{t-1}$. Since $q_1, \ldots, q_{t-1}$ form an orthonormal basis for the space spanned by $x_1, \ldots, x_{t-1}$, we have that $Q_{t-1}Q_{t-1}^T = \mathcal{P}_{t-1}$, the aforementioned orthogonal projection.

Since the first round of OHD is equivalent to a standard application of Hotelling's deflation, OHD inherits all of the weaknesses discussed in Section 2.2. However, the same principles may be applied to projection deflation to generate an orthogonalized variant that inherits its desirable properties.

Schur complement deflation is unique in that it preserves orthogonality in all subsequent rounds. That is, if a vector $v$ is orthogonal to $A_{t-1}$ for any $t$, then $A_t v = A_{t-1}v - \frac{A_{t-1}x_t x_t^T A_{t-1}v}{x_t^T A_{t-1}x_t} = 0$ as $A_{t-1}v = 0$. This further implies the following proposition.

**Proposition 2.2.** *Orthogonalized Schur complement deflation is equivalent to Schur complement deflation.*

*Proof.* Consider the $t$-th round of Schur complement deflation. We may write $x_t = o_t + p_t$, where $p_t$ is in the subspace spanned by all previously extracted pseudo-eigenvectors and $o_t$ is orthogonal to this subspace. Then we know that $A_{t-1}p_t = 0$, as $p_t$ is a linear combination of $x_1, \ldots, x_{t-1}$, and $A_{t-1}x_i = 0, \forall i < t$. Thus, $x_t^T A_t x_t = p_t^T A_t p_t + o_t^T A_t p_t + p_t^T A_t o_t + o_t^T A_t o_t = o_t^T A_t o_t$. Further, $A_{t-1}x_t x_t^T A_{t-1} = A_{t-1}p_t p_t^T A_{t-1} + A_{t-1}p_t o_t^T A_{t-1} + A_{t-1}o_t p_t^T A_{t-1} + A_{t-1}o_t o_t^T A_{t-1} = A_{t-1}o_t o_t^T A_{t-1}$. Hence, $A_t = A_{t-1} - \frac{A_{t-1}o_t o_t^T A_{t-1}}{o_t^T A_{t-1}o_t} = A_{t-1} - \frac{A_{t-1}q_t q_t^T A_{t-1}}{q_t^T A_{t-1}q_t}$ as $q_t = \frac{o_t}{||o_t||}$. $\qquad\square$

Table 1 compares the properties of the various deflation techniques studied in this section.

| Method | $x_t^T A_t x_t = 0$ | $A_t x_t = 0$ | $A_t \in \mathbb{S}_+^p$ | $A_s x_t = 0, \forall s > t$ |
|---|---|---|---|---|
| Hotelling's | ✓ | ✗ | ✗ | ✗ |
| Projection | ✓ | ✓ | ✓ | ✗ |
| Schur complement | ✓ | ✓ | ✓ | ✓ |
| Orth. Hotelling's | ✓ | ✗ | ✗ | ✗ |
| Orth. Projection | ✓ | ✓ | ✓ | ✓ |

Table 1: Summary of sparse PCA deflation method properties

## 3 Reformulating sparse PCA

In the previous section, we focused on heuristic deflation techniques that allowed us to reuse the cardinality-constrained optimization problem of Eq. (3). In this section, we explore a more principled alternative: reformulating the sparse PCA optimization problem to explicitly reflect our maximization objective on each round.

Recall that the goal of sparse PCA is to find $r$ cardinality-constrained pseudo-eigenvectors which together explain the most variance in the data. If we additionally constrain the sparse loadings to

be generated sequentially, as in the PCA setting and the previous section, then a greedy approach of maximizing the *additional* variance of each new vector naturally suggests itself.

On round $t$, the additional variance of a vector $x$ is given by $\frac{q^T A_0 q}{q^T q}$ where $A_0$ is the data covariance matrix, $q = (I - \mathcal{P}_{t-1})x$, and $\mathcal{P}_{t-1}$ is the projection onto the space spanned by previous pseudo-eigenvectors $x_1, \ldots, x_{t-1}$. As $q^T q = x^T(I - \mathcal{P}_{t-1})(I - \mathcal{P}_{t-1})x = x^T(I - \mathcal{P}_{t-1})x$, maximizing additional variance is equivalent to solving a cardinality-constrained maximum generalized eigenvalue problem,

$$\max_x \quad x^T(I - \mathcal{P}_{t-1})A_0(I - \mathcal{P}_{t-1})x$$
$$\text{subject to } x^T(I - \mathcal{P}_{t-1})x = 1 \tag{7}$$
$$\mathbf{Card}(x) \le k_t.$$

If we let $q_s = (I - \mathcal{P}_{s-1})x_s, \forall s \le t - 1$, then $q_1, \ldots, q_{t-1}$ form an orthonormal basis for the space spanned by $x_1, \ldots, x_{t-1}$. Writing $I - \mathcal{P}_{t-1} = I - \sum_{s=1}^{t-1} q_s q_s^T = \prod_{s=1}^{t-1}(I - q_s q_s^T)$ suggests a generalized deflation technique that leads to the solution of Eq. (7) on each round. We imbed the technique into the following algorithm for sparse PCA:

---

**Algorithm 1** Generalized Deflation Method for Sparse PCA

---

Given: $A_0 \in S_+^p$, $r \in \mathbb{N}$, $\{k_1, \ldots, k_r\} \subset \mathbb{N}$
Execute:

1. $B_0 \leftarrow I$
2. For $t := 1, \ldots, r$
    - $x_t \leftarrow \underset{x : x^T B_{t-1} x = 1, \mathbf{Card}(x) \le k_t}{\operatorname{argmax}} x^T A_{t-1} x$
    - $q_t \leftarrow B_{t-1} x_t$
    - $A_t \leftarrow (I - q_t q_t^T) A_{t-1} (I - q_t q_t^T)$
    - $B_t \leftarrow B_{t-1}(I - q_t q_t^T)$
    - $x_t \leftarrow x_t / \|x_t\|$

Return: $\{x_1, \ldots, x_r\}$

---

Adding a cardinality constraint to a maximum eigenvalue problem renders the optimization problem NP-hard [10], but any of several leading sparse eigenvalue methods, including GSLDA of [10], DCPCA of [12], and DSPCA of [1] (with a modified trace constraint), can be adapted to solve this cardinality-constrained generalized eigenvalue problem.

## 4    Experiments

In this section, we present several experiments on real world datasets to demonstrate the value added by our newly derived deflation techniques. We run our experiments with Matlab implementations of DCPCA [12] (with the continuity correction of [9]) and GSLDA [10], fitted with each of the following deflation techniques: Hotelling's (HD), projection (PD), Schur complement (SCD), orthogonalized Hotelling's (OHD), orthogonalized projection (OPD), and generalized (GD).

### 4.1    Pit props dataset

The pit props dataset [5] with 13 variables and 180 observations has become a de facto standard for benchmarking sparse PCA methods. To demonstrate the disparate behavior of differing deflation methods, we utilize each sparse PCA algorithm and deflation technique to successively extract six sparse loadings, each constrained to have cardinality less than or equal to $k_t = 4$. We report the additional variances explained by each sparse vector in Table 2 and the cumulative percentage variance explained on each iteration in Table 3. For reference, the first 6 true principal components of the pit props dataset capture 87% of the variance.

| DCPCA | | | | | | GSLDA | | | | | |
|---|---|---|---|---|---|---|---|---|---|---|---|
| HD | PD | SCD | OHD | OPD | GD | HD | PD | SCD | OHD | OPD | GD |
| 2.938 | 2.938 | 2.938 | 2.938 | 2.938 | 2.938 | 2.938 | 2.938 | 2.938 | 2.938 | 2.938 | 2.938 |
| 2.209 | 2.209 | 2.076 | 2.209 | 2.209 | 2.209 | 2.107 | 2.280 | 2.065 | 2.107 | 2.280 | 2.280 |
| 0.935 | 1.464 | 1.926 | 0.935 | 1.464 | 1.477 | 1.988 | 2.067 | 2.243 | 1.985 | 2.067 | 2.072 |
| 1.301 | 1.464 | 1.164 | 0.799 | 1.464 | 1.464 | 1.352 | 1.304 | 1.120 | 1.335 | 1.305 | 1.360 |
| 1.206 | 1.057 | 1.477 | 0.901 | 1.058 | 1.178 | 1.067 | 1.120 | 1.164 | 0.497 | 1.125 | 1.127 |
| 0.959 | 0.980 | 0.725 | 0.431 | 0.904 | 0.988 | 0.557 | 0.853 | 0.841 | 0.489 | 0.852 | 0.908 |

Table 2: Additional variance explained by each of the first 6 sparse loadings extracted from the Pit Props dataset.

On the DCPCA run, Hotelling's deflation explains 73.4% of the variance, while the best performing methods, Schur complement deflation and generalized deflation, explain approximately 79% of the variance each. Projection deflation and its orthogonalized variant also outperform Hotelling's deflation, while orthogonalized Hotelling's shows the worst performance with only 63.2% of the variance explained. Similar results are obtained when the discrete method of GSLDA is used. Generalized deflation and the two projection deflations dominate, with GD achieving the maximum cumulative variance explained on each round. In contrast, the more standard Hotelling's and orthogonalized Hotelling's underperform the remaining techniques.

| DCPCA | | | | | | GSLDA | | | | | |
|---|---|---|---|---|---|---|---|---|---|---|---|
| HD | PD | SCD | OHD | OPD | GD | HD | PD | SCD | OHD | OPD | GD |
| 22.6% | 22.6% | 22.6% | 22.6% | 22.6% | 22.6% | 22.6% | 22.6% | 22.6% | 22.6% | 22.6% | 22.6% |
| 39.6% | 39.6% | 38.6% | 39.6% | 39.6% | 39.6% | 38.8% | 40.1% | 38.5% | 38.8% | 40.1% | 40.1% |
| 46.8% | 50.9% | 53.4% | 46.8% | 50.9% | 51.0% | 54.1% | 56.0% | 55.7% | 54.1% | 56.0% | 56.1% |
| 56.8% | 62.1% | 62.3% | 52.9% | 62.1% | 62.2% | 64.5% | 66.1% | 64.4% | 64.3% | 66.1% | 66.5% |
| 66.1% | 70.2% | 73.7% | 59.9% | 70.2% | 71.3% | 72.7% | 74.7% | 73.3% | 68.2% | 74.7% | 75.2% |
| 73.4% | 77.8% | 79.3% | 63.2% | 77.2% | 78.9% | 77.0% | 81.2% | 79.8% | 71.9% | 81.3% | 82.2% |

Table 3: Cumulative percentage variance explained by the first 6 sparse loadings extracted from the Pit Props dataset.

## 4.2  Gene expression data

The Berkeley Drosophila Transcription Network Project (BDTNP) 3D gene expression data [4] contains gene expression levels measured in each nucleus of developing Drosophila embryos and averaged across many embryos and developmental stages. Here, we analyze 0-3_1160524183713_s10436-29ap05-02.vpc, an aggregate VirtualEmbryo containing 21 genes and 5759 example nuclei. We run GSLDA for eight iterations with cardinality pattern 9,7,6,5,3,2,2,2 and report the results in Table 4.

| | GSLDA additional variance explained | | | | | | GSLDA cumulative percentage variance | | | | | |
|---|---|---|---|---|---|---|---|---|---|---|---|---|
| | HD | PD | SCD | OHD | OPD | GD | HD | PD | SCD | OHD | OPD | GD |
| PC 1 | 1.784 | 1.784 | 1.784 | 1.784 | 1.784 | 1.784 | 21.0% | 21.0% | 21.0% | 21.0% | 21.0% | 21.0% |
| PC 2 | 1.464 | 1.453 | 1.453 | 1.464 | 1.453 | 1.466 | 38.2% | 38.1% | 38.1% | 38.2% | 38.1% | 38.2% |
| PC 3 | 1.178 | 1.178 | 1.179 | 1.176 | 1.178 | 1.187 | 52.1% | 51.9% | 52.0% | 52.0% | 51.9% | 52.2% |
| PC 4 | 0.716 | 0.736 | 0.716 | 0.713 | 0.721 | 0.743 | 60.5% | 60.6% | 60.4% | 60.4% | 60.4% | 61.0% |
| PC 5 | 0.444 | 0.574 | 0.571 | 0.460 | 0.571 | 0.616 | 65.7% | 67.4% | 67.1% | 65.9% | 67.1% | 68.2% |
| PC 6 | 0.303 | 0.306 | 0.278 | 0.354 | 0.244 | 0.332 | 69.3% | 71.0% | 70.4% | 70.0% | 70.0% | 72.1% |
| PC 7 | 0.271 | 0.256 | 0.262 | 0.239 | 0.313 | 0.304 | 72.5% | 74.0% | 73.4% | 72.8% | 73.7% | 75.7% |
| PC 8 | 0.223 | 0.239 | 0.299 | 0.257 | 0.245 | 0.329 | 75.1% | 76.8% | 77.0% | 75.9% | 76.6% | 79.6% |

Table 4: Additional variance and cumulative percentage variance explained by the first 8 sparse loadings of GSLDA on the BDTNP VirtualEmbryo.

The results of the gene expression experiment show a clear hierarchy among the deflation methods. The generalized deflation technique performs best, achieving the largest additional variance on every round and a final cumulative variance of 79.6%. Schur complement deflation, projection deflation, and orthogonalized projection deflation all perform comparably, explaining roughly 77% of the total variance after 8 rounds. In last place are the standard Hotelling's and orthogonalized Hotelling's deflations, both of which explain less than 76% of variance after 8 rounds.

## 5 Conclusion

In this work, we have exposed the theoretical and empirical shortcomings of Hotelling's deflation in the sparse PCA setting and developed several alternative methods more suitable for non-eigenvector deflation. Notably, the utility of these procedures is not limited to the sparse PCA setting. Indeed, the methods presented can be applied to any of a number of constrained eigendecomposition-based problems, including sparse canonical correlation analysis [13] and linear discriminant analysis [10].

**Acknowledgments**

This work was supported by AT&T through the AT&T Labs Fellowship Program.

## References

[1] A. d'Aspremont, L. El Ghaoui, M. I. Jordan, and G. R. G. Lanckriet. A Direct Formulation for Sparse PCA using Semidefinite Programming. In Advances in Neural Information Processing Systems (NIPS). Vancouver, BC, December 2004.

[2] A. d'Aspremont, F. R. Bach, and L. E. Ghaoui. Full regularization path for sparse principal component analysis. In Proceedings of the 24th international Conference on Machine Learning. Z. Ghahramani, Ed. ICML '07, vol. 227. ACM, New York, NY, 177-184, 2007.

[3] J. Cadima and I. Jolliffe. Loadings and correlations in the interpretation of principal components. Applied Statistics, 22:203.214, 1995.

[4] C.C. Fowlkes, C.L. Luengo Hendriks, S.V. Kernen, G.H. Weber, O. Rbel, M.-Y. Huang, S. Chatoor, A.H. DePace, L. Simirenko and C. Henriquez et al. Cell 133, pp. 364-374, 2008.

[5] J. Jeffers. Two case studies in the application of principal components. Applied Statistics, 16, 225-236, 1967.

[6] I.T. Jolliffe and M. Uddin. A Modified Principal Component Technique based on the Lasso. Journal of Computational and Graphical Statistics, 12:531.547, 2003.

[7] I.T. Jolliffe, Principal component analysis, Springer Verlag, New York, 1986.

[8] I.T. Jolliffe. Rotation of principal components: choice of normalization constraints. Journal of Applied Statistics, 22:29-35, 1995.

[9] B. Moghaddam, Y. Weiss, and S. Avidan. Spectral bounds for sparse PCA: Exact and greedy algorithms. Advances in Neural Information Processing Systems, 18, 2006.

[10] B. Moghaddam, Y. Weiss, and S. Avidan. Generalized spectral bounds for sparse LDA. In Proc. ICML, 2006.

[11] Y. Saad, Projection and deflation methods for partial pole assignment in linear state feedback, IEEE Trans. Automat. Contr., vol. 33, pp. 290-297, Mar. 1998.

[12] B.K. Sriperumbudur, D.A. Torres, and G.R.G. Lanckriet. Sparse eigen methods by DC programming. Proceedings of the 24th International Conference on Machine learning, pp. 831-838, 2007.

[13] D. Torres, B.K. Sriperumbudur, and G. Lanckriet. Finding Musically Meaningful Words by Sparse CCA. Neural Information Processing Systems (NIPS) Workshop on Music, the Brain and Cognition, 2007.

[14] P. White. The Computation of Eigenvalues and Eigenvectors of a Matrix. Journal of the Society for Industrial and Applied Mathematics, Vol. 6, No. 4, pp. 393-437, Dec., 1958.

[15] F. Zhang (Ed.). The Schur Complement and Its Applications. Kluwer, Dordrecht, Springer, 2005.

[16] Z. Zhang, H. Zha, and H. Simon, Low-rank approximations with sparse factors I: Basic algorithms and error analysis. SIAM J. Matrix Anal. Appl., 23 (2002), pp. 706-727.

[17] Z. Zhang, H. Zha, and H. Simon, Low-rank approximations with sparse factors II: Penalized methods with discrete Newton-like iterations. SIAM J. Matrix Anal. Appl., 25 (2004), pp. 901-920.

[18] H. Zou, T. Hastie, and R. Tibshirani. Sparse Principal Component Analysis. Technical Report, Statistics Department, Stanford University, 2004.

